# Implicit Differentiation by Perturbation

**Justin Domke**
Rochester Institute of Technology
`justin.domke@rit.edu`

## Abstract

This paper proposes a simple and efficient finite difference method for implicit differentiation of marginal inference results in discrete graphical models. Given an arbitrary loss function, defined on marginals, we show that the derivatives of this loss with respect to model parameters can be obtained by running the inference procedure twice, on slightly perturbed model parameters. This method can be used with approximate inference, with a loss function over approximate marginals. Convenient choices of loss functions make it practical to fit graphical models with hidden variables, high treewidth and/or model misspecification.

## 1   Introduction

As graphical models are applied to more complex problems, it is increasingly necessary to learn parameters from data. Though the likelihood and conditional likelihood are the most widespread training objectives, these are sometimes undesirable and/or infeasible in real applications.

With low treewidth, if the data is truly distributed according to the chosen graphical model with some parameters, any consistent loss function will recover those true parameters in the high-data limit, and so one might select a loss function according to statistical convergence rates [1]. In practice, the model is usually misspecified to some degree, meaning no "true" parameters exist. In this case, different loss functions lead to different asymptotic parameter estimates. Hence, it is useful to consider the priorities of the user when learning. For low-treewidth graphs, several loss functions have been proposed that prioritize different types of accuracy (section 2.2). For parameters $\boldsymbol{\theta}$, these loss functions are given as a function $L(\boldsymbol{\mu}(\boldsymbol{\theta}))$ of marginals $\boldsymbol{\mu}(\boldsymbol{\theta})$. One can directly calculate $\frac{\partial L}{\partial \boldsymbol{\mu}}$. The parameter gradient $\frac{dL}{d\boldsymbol{\theta}}$ can be efficiently computed by loss-specific message-passing schemes[2, 3].

The likelihood may also be infeasible to optimize, due to the computational intractability of computing the log-partition function or its derivatives in high treewidth graphs. On the other hand, if an approximate inference algorithm will be used at test time, it is logical to design the loss function to compensate for defects in inference. The surrogate likelihood (the likelihood with an approximate partition function) can give superior results to the true likelihood, when approximate inference is used at test time[4].

The goal of this paper is to efficiently fit parameters to optimize an arbitrary function of predicted marginals, in a high-treewidth setting. If $\boldsymbol{\mu}(\boldsymbol{\theta})$ is the function mapping parameters to (approximate) marginals, and there is some loss function $L(\boldsymbol{\mu})$ defined on those marginals, we desire to recover $\frac{dL}{d\boldsymbol{\theta}}$. This enables the use of the marginal-based loss functions mentioned previously, but defined on *approximate* marginals.

There are two major existing approaches for calculating $\frac{dL}{d\boldsymbol{\theta}}$. First, after performing inference, this gradient can be obtained by solving a large, sparse linear system[5]. The major disadvantage of this approach is that standard linear solvers can perform poorly on large

| True (**y**) | Noisy (**x**) | Surrogate likelihood | Clique likelihood | Univariate likelihood | Smooth class. error |
|---|---|---|---|---|---|

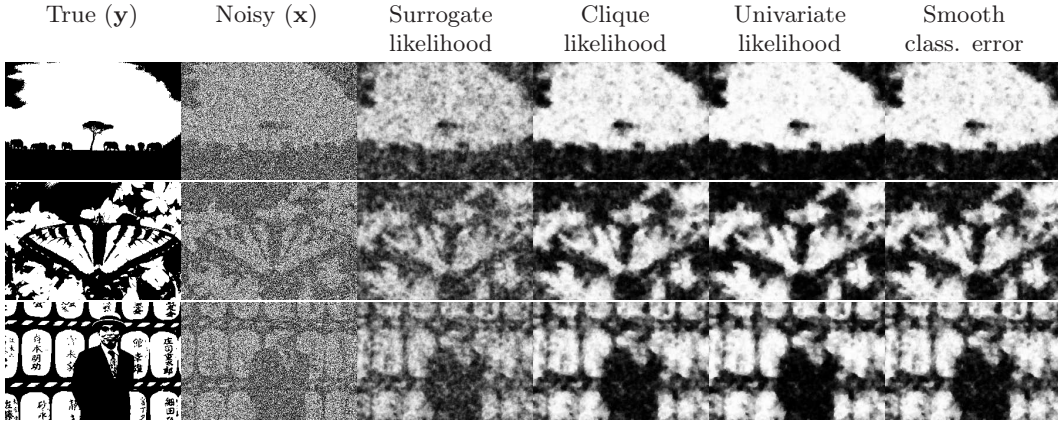

Figure 1: Example images from the Berkeley dataset, along with marginals for a conditional random field fit with various loss functions.

graphs, meaning that calculating this gradient can be more expensive than performing inference (Section 4). A second option is the Back Belief Propagation (BBP) algorithm[6]. This is based on application of reverse-mode automatic differentiation (RAD) to message passing. Crucially, this can be done without storing all intermediate messages, avoiding the enormous memory requirements of a naive application of RAD. This is efficient, with running-time in practice similar to inference. However, it is tied to a specific entropy approximation (Bethe) and algorithm (Loopy Belief Propagation). Extension to similar message-passing algorithms appears possible, but extension to more complex inference algorithms [7, 8, 9] is unclear.

Here, we observe that the loss gradient can be calculated by far more straightforward means. Our basic result is extremely simple: $\frac{dL}{d\boldsymbol{\theta}} \approx \frac{1}{r}\big(\boldsymbol{\mu}(\boldsymbol{\theta}+r\frac{\partial L}{\partial\boldsymbol{\mu}})-\boldsymbol{\mu}(\boldsymbol{\theta})\big)$, with equality in the limit $r \rightarrow 0$. This result follows from, first, the well-known trick of approximating Jacobian-vector products by finite differences and, second, the special property that for marginal inference, the Jacobian matrix $\frac{d\boldsymbol{\mu}}{d\boldsymbol{\theta}^T}$ is symmetric. This result applies when marginal inference takes place over the local polytope with an entropy that is concave and obeys a minor technical condition. It can also be used with non-concave entropies, with some assumptions on how inference recovers different local optima. It is easy to use this to compute the gradient of essentially any differentiable loss function defined on marginals. Effectively, all one needs to do is re-run the inference procedure on a set of parameters slightly "perturbed" in the direction $\frac{\partial L}{\partial\boldsymbol{\mu}}$. Conditional training and tied or nonlinear parameters can also be accommodated.

One clear advantage of this approach is simplicity and ease of implementation. Aside from this, like the matrix inversion approach, it is independent of the algorithm used to perform independence, and applicable to a variety of different inference approximations. Like BBP, the method is efficient in that it makes only two calls to inference.

## 2 Background

### 2.1 Marginal Inference

This section briefly reviews the aspects of graphical models and marginal inference that are required for the rest of the paper. Let **x** denote a vector of discrete random variables. We use the exponential family representation

$$p(\mathbf{x};\boldsymbol{\theta}) = \exp\big(\boldsymbol{\theta}\cdot\mathbf{f}(\mathbf{x})-A(\boldsymbol{\theta})\big), \tag{1}$$

where $\mathbf{f}(\mathbf{x})$ is the features of the observation **x**, and $A = \log\sum_{\mathbf{x}}\exp\boldsymbol{\theta}\cdot\mathbf{f}(\mathbf{x})$ assures normalization. For graphical models, $\mathbf{f}$ is typically a vector of indicator functions for each possible configuration of each factor and variable. With a slight abuse of set notation to represent

a vector, this can be written as $\mathbf{f}(\mathbf{x}) = \{I[\mathbf{x}_\alpha]\} \cup \{I[x_i]\}$. It is convenient to refer to the components of vectors like those in Eq. 1 using function notation. Write $\theta(\mathbf{x}_\alpha)$ to refer to the component of $\boldsymbol{\theta}$ corresponding to the indicator function $I[\mathbf{x}_\alpha]$, and similarly for $\theta(x_i)$. This gives an alternative representation for $p$, namely

$$p(\mathbf{x}; \boldsymbol{\theta}) = \exp\Big(\sum_\alpha \theta(\mathbf{x}_\alpha) + \sum_i \theta(x_i) - A(\boldsymbol{\theta})\Big). \tag{2}$$

Marginal inference means recovering the expected value of $\mathbf{f}$ or, equivalently, the marginal probability that each factor or variable have a particular value.

$$\boldsymbol{\mu}(\boldsymbol{\theta}) \quad = \quad \sum_\mathbf{x} p(\mathbf{x}; \boldsymbol{\theta}) \mathbf{f}(\mathbf{x}) \tag{3}$$

Though marginals could, in principle, be computed by the brute-force sum in Eq. 3, it is useful to consider the paired variational representation [10, Chapter 3]

$$A(\boldsymbol{\theta}) \quad = \quad \max_{\boldsymbol{\mu} \in \mathcal{M}} \boldsymbol{\theta} \cdot \boldsymbol{\mu} + H(\boldsymbol{\mu}) \tag{4}$$

$$\boldsymbol{\mu}(\boldsymbol{\theta}) = \frac{dA}{d\boldsymbol{\theta}} \quad = \quad \arg\max_{\boldsymbol{\mu} \in \mathcal{M}} \boldsymbol{\theta} \cdot \boldsymbol{\mu} + H(\boldsymbol{\mu}), \tag{5}$$

in which $A$ and $\boldsymbol{\mu}$ can both be recovered from solving the same optimization problem. Here, $\mathcal{M} = \{\boldsymbol{\mu}(\boldsymbol{\theta}) | \boldsymbol{\theta} \in \Re^n\}$ is the marginal polytope– those marginals $\boldsymbol{\mu}$ resulting from some parameter vector $\boldsymbol{\theta}$. Similarly, $H(\boldsymbol{\mu})$ is the entropy of $p(\mathbf{x}; \boldsymbol{\theta}')$, where $\boldsymbol{\theta}'$ is the vector of parameters that produces the marginals $\boldsymbol{\mu}$.

As $\mathcal{M}$ is a convex set, and $H$ a concave function, Eq. 5 is equivalent to a convex optimization problem. Nevertheless it is difficult to characterize $\mathcal{M}$ or compute $H(\boldsymbol{\mu})$ in high-treewidth graphs. A variety of approximate inference methods can be seen as solving a modification of Eqs. 4 and 5, with the marginal polytope and entropy replaced with tractable approximations. Notice that these are also paired; the approximate $\boldsymbol{\mu}$ is the *exact* gradient of the approximate $A$.

The commonest relaxation of $\mathcal{M}$ is the local polytope

$$\mathcal{L} = \{\boldsymbol{\mu} \geq \mathbf{0} \,|\, \mu(x_i) = \sum_{\mathbf{x}_{\alpha \setminus i}} \mu(\mathbf{x}_\alpha), \ \sum_{x_i} \mu(x_i) = 1\}. \tag{6}$$

This underlies loopy belief propagation, as well as tree-reweighted belief propagation. Since a valid set of marginals must obey these constraints, $\mathcal{L} \supseteq \mathcal{M}$. Note that since the equality constraints are linear, there exists a matrix $B$ and vector $\mathbf{d}$ such that

$$\mathcal{L} = \{\boldsymbol{\mu} \geq \mathbf{0} | B\boldsymbol{\mu} = \mathbf{d}\}. \tag{7}$$

A variety of entropy approximations exist. The Bethe approximation implicit in loopy belief propagation [11] is non-concave in general, which results in sometimes failing to achieve the global optimum. Concave entropy functions include the tree-reweighted entropy [12], convexified Bethe entropies [13], and the class of entropies obeying Heskes' conditions [14].

## 2.2   Loss Functions

Given some data, $\{\hat{\mathbf{x}}\}$, we will pick the parameters $\boldsymbol{\theta}$ to minimize the empirical risk

$$\sum_{\hat{\mathbf{x}}} L(\hat{\mathbf{x}}; \boldsymbol{\theta}). \tag{8}$$

**Likelihood.**   The (negative) likelihood is the classic loss function for training graphical models. Exploiting the fact that $dA/d\boldsymbol{\theta} = \boldsymbol{\mu}(\boldsymbol{\theta})$, the gradient is available in closed-form.

$$
\begin{aligned}
L(\hat{\mathbf{x}}; \boldsymbol{\theta}) &= -\log p(\hat{\mathbf{x}}; \boldsymbol{\theta}) \\
&= -\boldsymbol{\theta} \cdot \mathbf{f}(\hat{\mathbf{x}}) + A(\boldsymbol{\theta}). \quad (9)
\end{aligned}
\qquad
\frac{dL}{d\boldsymbol{\theta}} = -\mathbf{f}(\hat{\mathbf{x}}) + \boldsymbol{\mu}(\boldsymbol{\theta}). \qquad (10)
$$

**Surrogate Likelihood.** Neither $A$ nor $\boldsymbol{\mu}$ is tractable with high treewidth. However, if written in variational form (Eqs. 4 and 5), they can be approximated using approximate inference. The surrogate likelihood [4] is simply the likelihood as in Eq. 9 with an approximate $A$. It has the gradient as in Eq. 10, but with approximate marginals $\boldsymbol{\mu}$.

Unlike the losses below, the surrogate likelihood is convex when based on a concave inference method. See Ganapathi et al.[15] for a variant of this for inference with local optima.

**Univariate Likelihood.** If the application will only make use of univariate marginals at test time, one might fit parameters specifically to make these univariate marginals accurate. Kakade et al.[3] proposed the loss

$$
L(\hat{\mathbf{x}}; \boldsymbol{\theta}) = -\sum_i \log \mu(\hat{x}_i; \boldsymbol{\theta}). \qquad (11)
$$

This can be computed in treelike graphs, after running belief propagation to compute marginals. A message-passing scheme can efficiently compute the gradient.

**Univariate Classification Error.** Some applications only use the maximum probability marginals. Gross at al.[2] considered the loss

$$
L(\hat{\mathbf{x}}; \boldsymbol{\theta}) = \sum_i S\big( \max_{x_i \neq \hat{x}_i} \mu(x_i; \boldsymbol{\theta}) - \mu(\hat{x}_i; \boldsymbol{\theta}) \big), \qquad (12)
$$

where $S$ is the step function. This loss measures the number of incorrect components of $\hat{\mathbf{x}}$ if each is predicted to be the "max marginal". However, since this is non-differentiable, it is suggested to approximate this by replacing $S$ with a sigmoid function $S(t) = (1 + \exp(-\lambda t))^{-1}$, where $\lambda$ controls the approximation quality. Our experiments use $\lambda = 50$.

As with the univariate likelihood, this loss can be computed if exact marginals are available. Computing the gradient requires another message passing scheme.

**Clique loss functions.** One can easily define clique versions of the previous two loss functions, where the summations are over $\alpha$, rather than $i$. These measure the accuracy of clique-wise marginals, rather than univariate marginals.

## 2.3 Implicit Differentiation

As noted in Eq. 7, the equality constraints in the local polytope are linear, and hence when the positivity constraint can be disregarded, approximate marginal inference algorithms can be seen as solving the optimization $\boldsymbol{\mu}(\boldsymbol{\theta}) = \arg\max_{\boldsymbol{\mu}, B\boldsymbol{\mu}=\mathbf{d}} \boldsymbol{\theta} \cdot \boldsymbol{\mu} + H(\boldsymbol{\mu})$. Domke showed[5], in our notation, that

$$
\frac{dL}{d\boldsymbol{\theta}} = \big( D^{-1} B^T (B D^{-1} B^T)^{-1} B D^{-1} - D^{-1} \big) \frac{dL}{d\boldsymbol{\mu}}, \qquad (13)
$$

where $D = \frac{\partial^2 H}{\partial \boldsymbol{\mu} \partial \boldsymbol{\mu}^T}$ is the (diagonal) Hessian of the entropy approximation.

Unfortunately, this requires solving a sparse linear system for each training example and iteration. As we will see below, with large or poorly conditioned problems, the computational expense of this can far exceed that of inference. Note that $B D^{-1} B^T$ is, in general, indefinite, restricting what solvers can be used. Another limitation is that $D$ can be singular if any counting numbers (Eq. 16) are zero.

## 2.4 Conditional training and nonlinear parameters.

For simplicity, all the above discussion was confined to fully parametrized models. Nonlinear and tied parameters are easily dealt with by considering $\boldsymbol{\theta}(\boldsymbol{\phi})$ to be a function of the "true"

**Algorithm 1** Calculating loss derivatives (two-sided).

1. Do inference. $\boldsymbol{\mu}^* \leftarrow \arg\max_{\boldsymbol{\mu} \in \mathcal{M}} \boldsymbol{\theta} \cdot \boldsymbol{\mu} + H(\boldsymbol{\mu})$

2. At $\boldsymbol{\mu}^*$, calculate the partial derivative $\dfrac{\partial L}{\partial \boldsymbol{\mu}}$.

3. Calculate a perturbation size $r$.

4. Do inference on perturbed parameters.
$$\boldsymbol{\mu}^+ \leftarrow \arg\max_{\boldsymbol{\mu} \in \mathcal{M}} (\boldsymbol{\theta} + r\frac{\partial L}{\partial \boldsymbol{\mu}}) \cdot \boldsymbol{\mu} + H(\boldsymbol{\mu}) \qquad \boldsymbol{\mu}^- \leftarrow \arg\max_{\boldsymbol{\mu} \in \mathcal{M}} (\boldsymbol{\theta} - r\frac{\partial L}{\partial \boldsymbol{\mu}}) \cdot \boldsymbol{\mu} + H(\boldsymbol{\mu})$$

5. Recover full derivative. $\dfrac{dL}{d\boldsymbol{\theta}} \leftarrow \dfrac{1}{2r}(\boldsymbol{\mu}^+ - \boldsymbol{\mu}^-)$

parameters $\boldsymbol{\phi}$. Once $dL/d\boldsymbol{\theta}$ is known $dL/d\boldsymbol{\phi}$ can be recovered by a simple application of the chain rule, namely

$$\frac{dL}{d\boldsymbol{\phi}} = \frac{d\boldsymbol{\theta}^T}{d\boldsymbol{\phi}} \frac{dL}{d\boldsymbol{\theta}}. \tag{14}$$

Conditional training is similar: define a distribution over a random variable $\mathbf{y}$, parametrized by $\boldsymbol{\theta}(\boldsymbol{\phi}; \mathbf{x})$, the derivative on a particular pair $(\mathbf{x}, \mathbf{y})$ is given again by Eq. 14. Examples of both of these are in the experiments.

## 3   Implicit Differentiation by Perturbation

This section shows that when $\boldsymbol{\mu}(\boldsymbol{\theta}) = \arg\max_{\boldsymbol{\mu} \in \mathcal{L}} \boldsymbol{\theta} \cdot \boldsymbol{\mu} + H(\boldsymbol{\mu})$, the loss gradient can be computed by Alg. 1 for a concave entropy approximation of the form

$$H(\boldsymbol{\mu}) \;=\; -\sum_{\alpha} c_\alpha \sum_{\mathbf{x}_\alpha} \mu(\mathbf{x}_\alpha) \log \mu(\mathbf{x}_\alpha) - \sum_i c_i \sum_{x_i} \mu(x_i) \log \mu(x_i), \tag{15}$$

when the counting numbers $c$ obey (as is true of most proposed entropies)

$$c_\alpha > 0, \; c_i + \sum_{\alpha, i \in \alpha} c_\alpha > 0. \tag{16}$$

For intuition, the following Lemma uses notation $(\boldsymbol{\mu}, \boldsymbol{\theta}, H)$ suggesting the application to marginal inference. However, note that the result is true for any functions satisfying the stated conditions.

**Lemma.** *If $\boldsymbol{\mu}(\boldsymbol{\theta})$ is implicitly defined by*

$$\boldsymbol{\mu}(\boldsymbol{\theta}) \;=\; \arg\max_{\boldsymbol{\mu}} \boldsymbol{\mu} \cdot \boldsymbol{\theta} + H(\boldsymbol{\mu}) \tag{17}$$

$$s.t \qquad B\boldsymbol{\mu} - \mathbf{d} = \mathbf{0}, \tag{18}$$

*where $H(\boldsymbol{\mu})$ is strictly convex and twice differentiable, then $\dfrac{d\boldsymbol{\mu}}{d\boldsymbol{\theta}^T}$ exists and is symmetric.*

*Proof.* First, form a Lagrangian enforcing the constraints on the objective function.

$$\mathbb{L} = \boldsymbol{\mu} \cdot \boldsymbol{\theta} + H(\boldsymbol{\mu}) + \boldsymbol{\lambda}^T (B\boldsymbol{\mu} - \mathbf{d}) \tag{19}$$

The solution is $\boldsymbol{\mu}$ and $\boldsymbol{\lambda}$ such that $d\mathbb{L}/d\boldsymbol{\mu} = \mathbf{0}$ and $d\mathbb{L}/d\boldsymbol{\lambda} = \mathbf{0}$.

$$\begin{bmatrix} \boldsymbol{\theta} + \partial H(\boldsymbol{\mu})/\partial \boldsymbol{\mu} + B^T \boldsymbol{\lambda} \\ B\boldsymbol{\mu} - \mathbf{d} \end{bmatrix} = \begin{bmatrix} \mathbf{0} \\ \mathbf{0} \end{bmatrix} \tag{20}$$

Recall the general implicit function theorem. If $\mathbf{f}(\boldsymbol{\theta})$ is implicitly defined by the constraint that $\mathbf{h}(\boldsymbol{\theta}, \mathbf{f}) = \mathbf{0}$, then

$$\frac{d\mathbf{f}}{d\boldsymbol{\theta}^T} = -\Big(\frac{\partial \mathbf{h}}{\partial \mathbf{f}^T}\Big)^{-1} \frac{\partial \mathbf{h}}{\partial \boldsymbol{\theta}^T}. \tag{21}$$

Using Eq. 20 as our definition of $\mathbf{h}$, and differentiating with respect to both $\boldsymbol{\mu}$ and $\boldsymbol{\lambda}$, we have

$$\left[\begin{array}{c} d\boldsymbol{\mu}/d\boldsymbol{\theta}^T \\ d\boldsymbol{\lambda}/d\boldsymbol{\theta}^T \end{array}\right] = -\left[\begin{array}{cc} \partial^2 H/\partial\boldsymbol{\mu}\partial\boldsymbol{\mu}^T & B \\ B^T & 0 \end{array}\right]^{-1}\left[\begin{array}{c} I \\ 0 \end{array}\right]. \tag{22}$$

We see that $-d\boldsymbol{\mu}/d\boldsymbol{\theta}^T$ is the upper left block of the matrix being inverted. The result follows, since the inverse of a symmetric matrix is symmetric. $\qquad\square$

The following is the main result driving this paper. Again, this uses notation suggesting the application to implicit differentiation and marginal inference, but holds true for any functions satisfying the stated conditions.

**Theorem.** *Let $\boldsymbol{\mu}(\boldsymbol{\theta})$ be defined as in the previous Lemma, and let $L(\boldsymbol{\theta})$ be defined by $L(\boldsymbol{\theta}) = M\big(\boldsymbol{\mu}(\boldsymbol{\theta})\big)$ for some differentiable function $M(\boldsymbol{\mu})$. Then the derivative of $L$ with respect to $\boldsymbol{\theta}$ is given by*

$$\frac{dL}{d\boldsymbol{\theta}} = \lim_{r \to 0} \frac{1}{r}\Big(\boldsymbol{\mu}(\boldsymbol{\theta} + r\frac{\partial M}{\partial \boldsymbol{\mu}}) - \boldsymbol{\mu}(\boldsymbol{\theta})\Big). \tag{23}$$

*Proof.* First note that, by the vector chain rule,

$$\frac{dL}{d\boldsymbol{\theta}} = \frac{d\boldsymbol{\mu}^T}{d\boldsymbol{\theta}}\frac{\partial M}{\partial \boldsymbol{\mu}}. \tag{24}$$

Next, take some vector $\mathbf{v}$. By basic calculus, the derivative of $\boldsymbol{\mu}(\boldsymbol{\theta})$ in the direction of $\mathbf{v}$ is

$$\frac{d\boldsymbol{\mu}}{d\boldsymbol{\theta}^T}\mathbf{v} = \lim_{r \to 0} \frac{1}{r}\big(\boldsymbol{\mu}(\boldsymbol{\theta} + r\mathbf{v}) - \boldsymbol{\mu}(\boldsymbol{\theta})\big). \tag{25}$$

The result follows from substituting $\partial M/\partial \boldsymbol{\mu}$ for $\mathbf{v}$, and using the previous lemma to establish that $d\boldsymbol{\mu}/d\boldsymbol{\theta}^T = d\boldsymbol{\mu}^T/d\boldsymbol{\theta}$. $\qquad\square$

Alg. 1 follows from applying this theorem to marginal inference. However, notice that this does not enforce the constraint that $\boldsymbol{\mu} \geq 0$. The following gives mild technical conditions under which $\boldsymbol{\mu}$ will be strictly positive, and so the above theorem applies.

**Theorem.** *If $H(\boldsymbol{\mu}) = \sum_\alpha c_\alpha H(\boldsymbol{\mu}_c) + \sum_i c_i H(\boldsymbol{\mu}_i)$, and $\boldsymbol{\mu}^*$ is a (possibly local) maximum of $\boldsymbol{\theta} \cdot \boldsymbol{\mu} + H(\boldsymbol{\mu})$, under the local polytope $\mathcal{L}$, then*

$$c_\alpha > 0,\ c_i + \sum_{\alpha, i \in \alpha} c_\alpha > 0 \longrightarrow \boldsymbol{\mu}^* > \mathbf{0}. \tag{26}$$

This is an extension of a previous result [11, Theorem 9] for the Bethe entropy. However, extremely minor changes to the existing proof give this stronger result.

Most proposed entropies satisfy these conditions, including the Bethe entropy ($c_\alpha = 1, c_i + \sum_{\alpha, i \in \alpha} c_\alpha = 1$), the TRW entropy ($c_\alpha = \rho(\alpha)$, $c_i + \sum_{\alpha, i \in \alpha} c_\alpha = 1$, where $\rho(\alpha) > 0$ is the probability that $\alpha$ appears in a randomly chosen tree) and any entropy satisfying the slightly strengthened versions on Heskes' conditions [14, 16, Section 2].

What about non-concave entropies? The only place concavity was used above was in establishing that Eq. 20 has a unique solution. With a non-concave entropy this condition is still *valid*, not not unique, since there can be local optima. BBP essentially calculates this

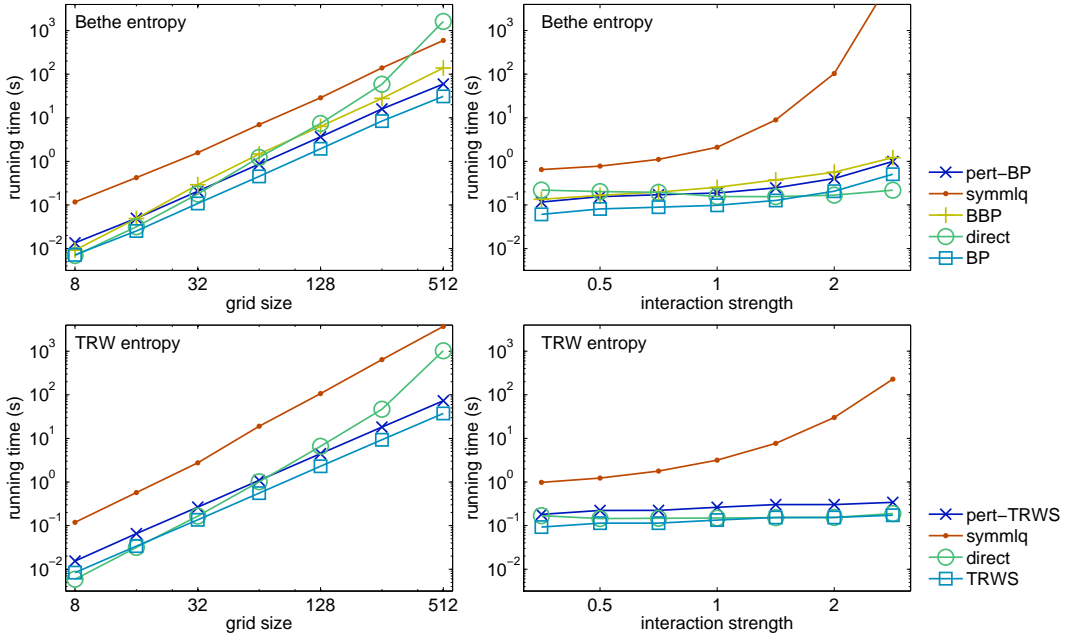

Figure 2: Times to compute $dL/d\boldsymbol{\theta}$ by perturbation, Back Belief Propagation (BBP), sparse matrix factorization (direct) and the iterative symmetric-LQ method (symmlq). Inference with BP and TRWS are shown for reference. As these results use two-sided differences, perturbation always takes twice the running time of the base inference algorithm. BBP takes time similar BP. Results use a pairwise grid with $x_i \in \{1, 2, ..., 5\}$, with univariate terms $\theta(x_i)$ taken uniformly from $[-1, +1]$ and interaction strengths $\theta(x_i, x_j)$ from $[-a, +a]$ for varying $a$. **Top Left**: Bethe entropy for varying grid sizes, with $a = 1$. Matrix factorization is efficient on small problems, but scales poorly. **Top Right**: Bethe entropy with a grid size of 32 and varying interaction strengths $a$. High interactions strengths lead to poor conditioning, slowing iterative methods. **Bottom Left**: Varying grid sizes with the TRW entropy. **Bottom Right**: TRW entropy with a grid size of 32 and varying interactions.

derivative by "tracking" the local optima. If perturbed beliefs are calculated from constant initial messages with a small step, one obtains the same result. Thus, BBP and perturbation give the same gradient for the Bethe approximation. (This was also verified experimentally.)

It remains to select the perturbation size $r$. Though the gradient is exact in the limit $r \to 0$, numerical error eventually dominates. Following Andrei[17], the experiments here use $r = \sqrt{\epsilon}(1 + |\boldsymbol{\theta}|_\infty)/|\frac{\partial L}{\partial \boldsymbol{\mu}}|_\infty$, where $\epsilon$ is machine epsilon.

## 4 Experiments

For inference, we used either loopy belief propagation, or tree-reweighted belief propagation. As these experiments take place on grids, we are able to make use of the convergent TRWS algorithm [18, Alg. 5], which we found to converge significantly faster than standard TRW. BP/TRWS were iterated until predicted beliefs changed less than $10^{-5}$ between iterations. BBP used a slightly looser convergence threshold of $10^{-4}$, which was similarly accurate.

Base code was implemented in Python, with C++ extensions for inference algorithms for efficiency. Sparse systems were solved directly using an interface to Matlab, which calls LAPACK. We selected the Symmetric LQ method as an iterative solver. Both solvers were the fastest among several tested on these problems. (Recall, the system is indefinite.) BBP results were computed by interfacing to the authors' implementation included in the libDAI toolkit[19]. We found the `PAR` mode, based on parallel updates [6, Eqs. 14-25] to be much slower than the more sophisticated `SEQ_FIX` mode, based on sequential updates [6, extended

Table 1: Binary denoising results, comparing the surrogate likelihood against three loss functions fit by implicit differentiation. All loss functions are per-pixel, based on tree-reweighted belief propagation with edge inclusion probabilities of .5. The "Best Published" results are the lowest previously reported pixelwise test errors using essentially loopy-belief propagation based surrogate likelihood. (For all losses, lower is better.)

| | Bimodal | Gaussian | Berkeley Segmentation Data | | | |
|---|---|---|---|---|---|---|
| **Test Loss** | Class. Error | Class. Error | Surrogate likelihood | Clique likelihood | Univariate likelihood | Class. Error |
| **Training Loss** | Train Test | Train Test | Train Test | Train Test | Train Test | Train Test |
| Surrogate likelihood | .0498 .0540 | .0286 .0239 | .251 .252 | 1.328 1.330 | .417 .416 | .141 .140 |
| Clique likelihood | .0488 .0535 | .0278 .0236 | .275 .277 | 1.176 1.178 | .316 .315 | .127 .126 |
| Univariate likelihood | .0493 .0541 | .0278 .0235 | .301 .303 | 1.207 1.210 | .305 .305 | .128 .127 |
| Smooth Class. Error | .0460 .0527 | .0273 .0241 | .281 .283 | 1.179 1.181 | .311 .310 | .127 .126 |
| Best Published [20] | .0548 | .0251 | | | | |

version, Fig. 5]. Hence, all results here use the latter. Other modes exceeded the available 12 GB memory. All experiments use a single core of a 2.26 GHz machine.

Our first experiment makes use of synthetically generated grid models. This allows systematic variance of graph size and parameter strength. With the TRW entropy, we use uniform edge appearance probabilities of $\rho = .49$, to avoid singularity in $D$. Our results (Fig. 2) can be summarized as follows. Matrix inversion (Eq. 13) with a direct solver is very efficient on small problems, but scales poorly. The iterative solver is expensive, and extremely sensitive to conditioning. With the Bethe approximation, perturbation performs similarly to BBP. TRWS converges faster than BP on poorly conditioned problems.

The second experiment considers a popular dataset for learning in high-treewidth graphical models[21]. This consists of four base images, each corrupted with 50 random noise patterns (either Gaussian or bimodal). Following the original work, 10 corrupted versions of the first base image are used for training, and the remaining 190 for testing. This dataset has been used repeatedly [22, 23], though direct comparison is sometimes complicated by varying model types and training/test set divisions. This experiment uses a grid model over neighboring pairs $(i, j)$

$$p(\mathbf{y}|\mathbf{x}) = \exp\Big(\sum_{i,j} \theta(y_i, y_j) + \sum_i \theta(y_i; x_i) - A(\boldsymbol{\theta}(\mathbf{x}))\Big), \tag{27}$$

where $\boldsymbol{\theta}(\mathbf{x})$ is a function of the input, with $\theta(y_i, y_j) = a(y_i, y_j)$ fully parametrized (independent of $\mathbf{x}$) and $\theta(y_i; x_i) = b(y_i)x_i + c(y_i)$ an affine function of $x_i$. Enforcing translation invariance gives a total of eight free parameters: four for $a(y_i, y_j)$, and two for $b(y_i)$, and $c(y_i)$[1]. Once $\frac{dL}{d\boldsymbol{\theta}}$ is known, we can, following Eq. 14, recover derivatives with respect to tied parameters[2].

Because the previous dataset is quite limited (only four base 64x64 images), all methods perform relatively well. Hence, we created a larger and more challenging dataset, consisting of 200 200x300 images from the Berkeley segmentation dataset, split half for training and testing. These are binarized by setting $y_i = 1$ if a pixel is above the image mean, and $y_i = 0$ otherwise. The noisy values $x_i$ are created by setting $x_i = y_i(1 - t_i^{1.25}) + (1 - y_i)t_i^{1.25}$, for $t_i$ uniform on $[0, 1]$.

Table 1 shows results for all three datasets. All the results below use batch L-BFGS for learning, and uniform edge appearance probabilities of $\rho = .5$. The surrogate likelihood performs well, in fact beating the best reported results on the bimodal and Gaussian data. However, the univariate and clique loss functions provide better univariate accuracy. Fig. 1 shows example results. The surrogate likelihood (which is convex), was used to initialize the univariate and clique likelihood, while the univariate likelihood was used to initialize the smooth classification error.

## Footnotes

[1]There are two redundancies, as adding a constant to $a(y_i, y_j)$ or $c(y_i)$ has no effect on $p$.

[2]Specifically, $\frac{dL}{da(y,y')} = \sum_{(i,j)} \frac{dL}{d\theta(y_i=y, y_j=y')}$, $\quad \frac{dL}{db(y)} = \sum_i \frac{dL}{d\theta(y_i=y)}x_i$, and $\frac{dL}{dc(y)} = \sum_i \frac{dL}{d\theta(y_i=y)}$.

# References

[1] Percy Liang and Michael Jordan. An asymptotic analysis of generative, discriminative, and pseudolikelihood estimators. In *ICML*, 2008.

[2] Samuel Gross, Olga Russakovsky, Chuong Do, and Serafim Batzoglou. Training conditional random fields for maximum labelwise accuracy. In *NIPS*. 2006.

[3] Sham Kakade, Yee Whye Teh, and Sam Roweis. An alternate objective function for Markovian fields. In *ICML*, 2002.

[4] Martin Wainwright. Estimating the "wrong" graphical model: Benefits in the computation-limited setting. *Journal of Machine Learning Research*, 7:1829–1859, 2006.

[5] Justin Domke. Learning convex inference of marginals. In *UAI*, 2008.

[6] Frederik Eaton and Zoubin Ghahramani. Choosing a variable to clamp. In *AISTATS*, 2009.

[7] Max Welling and Yee Whye Teh. Belief optimization for binary networks: A stable alternative to loopy belief propagation. In *UAI*, 2001.

[8] Tom Heskes, Kees Albers, and Bert Kappen. Approximate inference and constrained optimization. In *UAI*, 2003.

[9] Alan Yuille. CCCP algorithms to minimize the Bethe and Kikuchi free energies: Convergent alternatives to belief propagation. *Neural Computation*, 14:2002, 2002.

[10] Martin Wainwright and Michael Jordan. Graphical models, exponential families, and variational inference. *Found. Trends Mach. Learn.*, 1(1-2):1–305, 2008.

[11] Jonathan Yedidia, William Freeman, and Yair Weiss. Constructing free energy approximations and generalized belief propagation algorithms. *IEEE Transactions on Information Theory*, 51:2282–2312, 2005.

[12] Martin Wainwright, Tommi Jaakkola, and Alan Willsky. A new class of upper bounds on the log partition function. *IEEE Transactions on Information Theory*, 51(7):2313–2335, 2005.

[13] Ofer Meshi, Ariel Jaimovich, Amir Globerson, and Nir Friedman. Convexifying the bethe free energy. In *UAI*, 2009.

[14] Tom Heskes. Convexity arguments for efficient minimization of the bethe and kikuchi free energies. *J. Artif. Intell. Res. (JAIR)*, 26:153–190, 2006.

[15] Varun Ganapathi, David Vickrey, John Duchi, and Daphne Koller. Constrained approximate maximum entropy learning of markov random fields. In *UAI*, 2008.

[16] Tamir Hazan and Amnon Shashua. Convergent message-passing algorithms for inference over general graphs with convex free energies. In *UAI*, pages 264–273, 2008.

[17] Neculai Andrei. Accelerated conjugate gradient algorithm with finite difference hessian/vector product approximation for unconstrained optimization. *J. Comput. Appl. Math.*, 230(2):570–582, 2009.

[18] Talya Meltzer, Amir Globerson, and Yair Weiss. Convergent message passing algorithms - a unifying view, 2009.

[19] Joris M. Mooij et al. libDAI 0.2.4: A free/open source C++ library for Discrete Approximate Inference. http://www.libdai.org/, 2010.

[20] Sanjiv Kumar, Jonas August, and Martial Hebert. Exploiting inference for approximate parameter learning in discriminative fields: An empirical study. In *EMMCVPR*, 2005.

[21] Sanjiv Kumar and Martial Hebert. Discriminative random fields. *International Journal of Computer Vision*, 68(2):179–201, 2006.

[22] S. V. N. Vishwanathan, Nicol Schraudolph, Mark Schmidt, and Kevin Murphy. Accelerated training of conditional random fields with stochastic gradient methods. In *ICML*, 2006.

[23] Patrick Pletscher, Cheng Soon Ong, and Joachim Buhmann. Spanning tree approximations for conditional random fields. In *AISTATS*, 2009.

